# Scalable Inference of Overlapping Communities

**Prem Gopalan   David Mimno   Sean M. Gerrish   Michael J. Freedman   David M. Blei**

{pgopalan,mimno,sgerrish,mfreed,blei}@cs.princeton.edu
Department of Computer Science
Princeton University
Princeton, NJ 08540

## Abstract

We develop a scalable algorithm for posterior inference of overlapping communities in large networks. Our algorithm is based on stochastic variational inference in the mixed-membership stochastic blockmodel (MMSB). It naturally interleaves subsampling the network with estimating its community structure. We apply our algorithm on ten large, real-world networks with up to 60,000 nodes. It converges several orders of magnitude faster than the state-of-the-art algorithm for MMSB, finds hundreds of communities in large real-world networks, and detects the true communities in 280 benchmark networks with equal or better accuracy compared to other scalable algorithms.

## 1   Introduction

A central problem in network analysis is to identify communities, groups of related nodes with dense internal connections and few external connections [1, 2, 3]. Classical methods for community detection assume that each node participates in a single community [4, 5, 6]. This assumption is limiting, especially in large real-world networks. For example, a member of a large social network might belong to overlapping communities of co-workers, neighbors, and school friends.

To address this problem, researchers have developed several methods for detecting overlapping communities in observed networks. These methods include algorithmic approaches [7, 8] and probabilistic models [2, 3, 9, 10]. In this paper, we focus on the mixed-membership stochastic blockmodel (MMSB) [2], a probabilistic model that allows each node of a network to exhibit a mixture of communities. The MMSB casts community detection as posterior inference: Given an observed network, we estimate the posterior community memberships of its nodes.

The MMSB can capture complex community structure and has been adapted in several ways [11, 12]; however, its applications have been limited because its corresponding inference algorithms have not scaled to large networks [2]. In this work, we develop algorithms for the MMSB that scale, allowing us to study networks that were previously out of reach for this model. For example, we analyzed social networks with as many as 60,000 nodes. With our method, we can use the MMSB to analyze large networks, finding approximate posteriors in minutes with networks for which the original algorithm takes hours. When compared to other scalable methods for overlapping community detection, we found that the MMSB gives better predictions of new connections and more closely recovers ground-truth communities. Further, we can now use the MMSB to compute descriptive statistics at scale, such as which nodes bridge communities.

The original MMSB algorithm optimizes the variational objective by coordinate ascent, processing every pair of nodes in each iteration [2]. This algorithm is inefficient, and it quickly becomes intractable for large networks. In this paper, we develop stochastic optimization algorithms [13, 14] to fit the variational distribution, where we obtain noisy estimates of the gradient by subsampling the network.

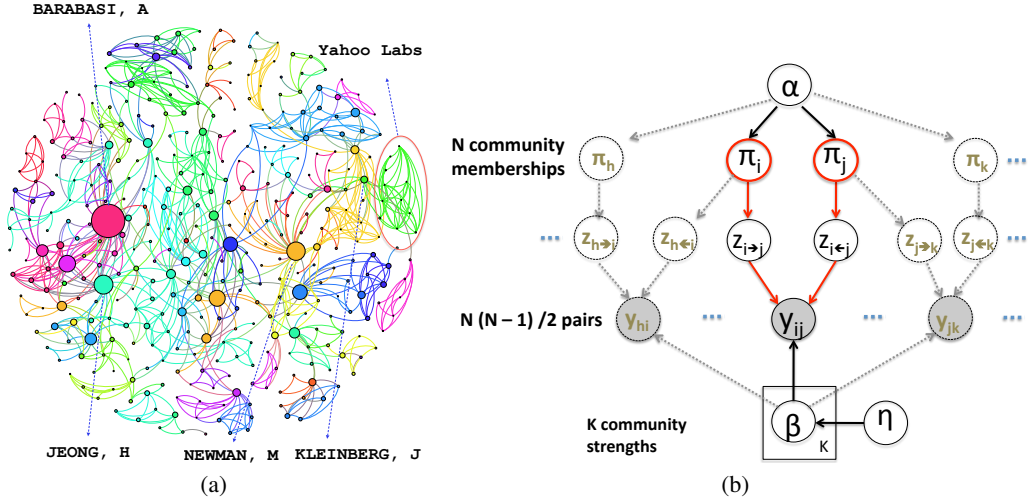

(a)           (b)

Figure 1: Figure 1(a) shows communities (see §2) discovered in a co-authorship network of 1,600 researchers [16] by an a-MMSB model with 50 communities. The color of author nodes indicates their most likely posterior community membership. The size of nodes indicates *bridgeness* [17], a measure of participation in multiple communities. Figure 1(b) shows a graphical model of the a-MMSB. The prior over multinomial $\pi$ is a symmetric Dirichlet distribution. Priors over Bernoulli $\boldsymbol{\beta}$ are Beta distributions.

Our algorithm alternates between subsampling from the network and adjusting its estimate of the underlying communities. While this strategy has been used in topic modeling [15], the MMSB introduces new challenges because the Markov blanket of each node is much larger than that of a document. Our simple sampler usually selects unconnected nodes (due to sparse real-world networks). We develop better sampling methods that focus more on the informative data in the network, e.g., the observed links, and thus make inference even faster.

## 2 Modeling overlapping communities

In this section, we introduce the assortative mixed-membership stochastic blockmodel (a-MMSB), a statistical model of networks that models nodes participating in multiple communities. The a-MMSB is a subclass of the mixed-membership stochastic blockmodel (MMSB) [2].[1]

Let $y$ denote the observed links of an undirected network, where $y_{ab} = 1$ if nodes $a$ and $b$ are linked and $0$ otherwise. Let $K$ denote the number of communities. Each node $a$ is associated with *community memberships* $\pi_a$, a distribution over communities; each community is associated with a *community strength* $\beta_k \in (0, 1)$, which captures how tightly its members are linked. The probability that two nodes are linked is governed by the similarity of their community memberships and the strength of their shared communities.

We capture these assumptions in the following generative process of a network.

1. For each community $k$, draw community strength $\beta_k \sim \text{Beta}(\eta)$.

2. For each node $a$, draw community memberships $\pi_a \sim \text{Dirichlet}(\alpha)$.

3. For each pair of nodes $a$ and $b$,

    (a) Draw interaction indicator $z_{a \to b} \sim \pi_a$.

    (b) Draw interaction indicator $z_{a \leftarrow b} \sim \pi_b$.

    (c) Draw link $y_{ab} \sim \text{Bernoulli}(r)$, where

$$r = \begin{cases} \beta_k & \text{if } z_{a \to b, k} = z_{a \leftarrow b, k} = 1, \\ \epsilon & \text{if } z_{a \to b} \neq z_{a \leftarrow b}. \end{cases} \tag{1}$$

Figure 1(b) represents the corresponding joint distribution of hidden and observed variables. The a-MMSB defines a single parameter $\epsilon$ to govern inter-community links. This captures assortativity—if two nodes are linked, it is likely that the latent community indicators were the same.

The full MMSB differs from the a-MMSB in that the former uses one parameter for each of the $K^2$ ordered pairs of communities. When the full MMSB is applied to undirected networks, two hypotheses compete to explain a link between each pair of nodes: either both nodes exhibit the same community or they are in different communities that link to each other.

We analyze data with a-MMSB via the posterior distribution over latent variables $p(\pi_{1:N}, \boldsymbol{z}, \beta_{1:K}|\boldsymbol{y}, \alpha, \eta)$. The posterior lets us form a predictive distribution of unseen links and measure latent network properties of the observed data. The posterior over $\pi_{1:N}$ represents the community memberships of the nodes, and the posterior over the interaction indicator variables $\boldsymbol{z}$ identifies *link communities* in the network [8]. For example, in a social network one member's link to another might arise because they are from the same high school while another might arise because they are co-workers. With an estimate of this latent structure, we can characterize the network in interesting ways. In Figure 1(a), we sized author nodes according to their expected posterior *bridgeness* [17], a measure of participation in multiple communities (see §5).

## 3 Stochastic variational inference

Our goal is to compute the posterior distribution $p(\pi_{1:N}, \boldsymbol{z}, \beta_{1:K}|\boldsymbol{y}, \alpha, \eta)$. Exact inference is intractable, so we use variational inference [18]. Traditional variational inference is a coordinate ascent algorithm. In the context of the MMSB (and the a-MMSB), coordinate ascent iterates between analyzing all $O(N^2)$ node pairs and updating the community memberships of the $N$ nodes [2]. In this section, we will derive a stochastic variational inference algorithm. Our algorithm iterates between sampling random pairs of nodes and updating node memberships. This avoids the per-iteration $O(N^2)$ computation and allows us to scale to large networks.

### 3.1 Variational inference in a-MMSB

In variational inference, we define a family of distributions over the hidden variables $q(\boldsymbol{\beta}, \boldsymbol{\pi}, \boldsymbol{z})$ and find the member of that family that is closest to the true posterior. (Closeness is measured with KL divergence.) We use the mean-field family, under which each variable is endowed with its own distribution and its own variational parameter. This allows us to tractably optimize the parameters to find a local minimum of the KL divergence. For the a-MMSB, the variational distributions are

$$q(z_{a \to b} = k) = \phi_{a \to b, k}; \quad q(\pi_a) = \text{Dirichlet}(\pi_a; \gamma_p); \quad q(\beta_k) = \text{Beta}(\beta_k; \lambda_k). \tag{2}$$

The posterior over link community assignments $\boldsymbol{z}$ is parameterized by the *per-interaction memberships* $\boldsymbol{\phi}$, the node community distributions $\boldsymbol{\pi}$ by the *community memberships* $\boldsymbol{\gamma}$, and the link probability $\boldsymbol{\beta}$ by the *community strengths* $\boldsymbol{\lambda}$. Notice that $\boldsymbol{\lambda}$ is of dimension $K \times 2$, and $\boldsymbol{\gamma}$ is of dimension $N \times K$.

Minimizing the KL divergence between $q$ and the true posterior is equivalent to optimizing an *evidence lower bound* (ELBO) $\mathcal{L}$, a bound on the log likelihood of the observations. We obtain this bound by applying Jensen's inequality [18] to the data likelihood. The ELBO is

$$\log p(\mathbf{y}|\alpha, \boldsymbol{\beta}) \geq \mathcal{L}(\boldsymbol{y}, \boldsymbol{\phi}, \boldsymbol{\gamma}, \boldsymbol{\lambda}) \triangleq \mathbb{E}_q[\log p(\mathbf{y}, \boldsymbol{\pi}, \boldsymbol{z}, \boldsymbol{\beta}|\alpha, \boldsymbol{\eta})] - \mathbb{E}_q[\log q(\boldsymbol{\beta}, \boldsymbol{\pi}, \boldsymbol{z})]. \tag{3}$$

The right side of Eq. 3 factorizes to

$$
\begin{aligned}
\mathcal{L} = &\sum_k \mathbb{E}_q[\log p(\beta_k|\eta_k)] - \sum_k \mathbb{E}_q[\log q(\beta_k|\lambda_k)] + \sum_n \mathbb{E}_q[\log p(\pi_n|\alpha)] - \sum_n \mathbb{E}_q[\log q(\pi_n|\gamma_n)] \\
&+ \sum_{a,b} \mathbb{E}_q[\log p(z_{a \to b}|\pi_a)] + \mathbb{E}_q[\log p(z_{a \leftarrow b}|\pi_b)] \\
&- \sum_{a,b} \mathbb{E}_q[\log q(z_{a \to b}|\phi_{a \to b})] - \mathbb{E}_q[\log q(z_{a \leftarrow b}|\phi_{a \leftarrow b})] \\
&+ \sum_{a,b} \mathbb{E}_q[\log p(y_{ab}|z_{a \to b}, z_{a \leftarrow b}, \boldsymbol{\beta})]
\end{aligned}
\tag{4}
$$

Notice the first line in Eq. 4 contains summations over communities and nodes; we call these *global terms*. They relate to the *global variables*, which are the community strengths $\boldsymbol{\lambda}$ and per-node memberships $\boldsymbol{\gamma}$. The remaining lines contain summations over all node pairs, which we call *local terms*. They depend on both the global and *local variables*, the latter being the per-interaction memberships $\boldsymbol{\phi}$. This distinction is important in the stochastic optimization algorithm.

## 3.2 Stochastic optimization

Our goal is to develop a variational inference algorithm that scales to large networks. We will use stochastic variational inference [14], which optimizes the ELBO with respect to the global variational parameters using stochastic gradient ascent. Stochastic gradient algorithms follow noisy estimates of the gradient with a decreasing step-size. If the expectation of the noisy gradient is equal to the gradient and if the step-size decreases according to a certain schedule, then we are guaranteed convergence to a local optimum [13]. Subsampling the data to form noisy gradients scales inference as we avoid the expensive all-pairs sums in Eq. 4.

Stochastic variational inference is a coordinate ascent algorithm that iteratively updates local and global parameters. For each iteration, we first subsample the network and compute optimal local parameters for the sample, given the current settings of the global parameters. We then update the global parameters using a stochastic natural gradient[2] computed from the subsampled data and local parameters. We call the first phase the local step and the second phase the global step [14].

The selection of subsamples in each iteration provides a way to plug in a variety of network subsampling algorithms. However, to maintain a correct stochastic optimization algorithm of the variational objective, the subsampling method must be valid. That is, the natural gradients estimated from the subsample must be unbiased estimates of the true gradients.

**The global step.** The global step updates the global community strengths $\boldsymbol{\lambda}$ and community memberships $\boldsymbol{\gamma}$ with a stochastic gradient of the ELBO in Eq. 4. Eq. 4 contains summations over all $O(N^2)$ node pairs. Now consider drawing a node pair $(a, b)$ at random from a population distribution $g(a, b)$ over the $M = N(N-1)/2$ node pairs. We can rewrite the ELBO as a random function of the variational parameters that includes the global terms and the local terms associated only with $(a, b)$. The expectation of this random function is equal in objective to Eq. 4. For example, the fourth term in Eq. 4 is rewritten as

$$\sum_{a,b} \mathbb{E}_q[\log p(y_{ab}|z_{a\to b}, z_{a\leftarrow b}, \boldsymbol{\beta})] = \mathbb{E}_g[\frac{1}{g(a,b)}\mathbb{E}_q[\log p(y_{ab}|z_{a\to b}, z_{a\leftarrow b}, \boldsymbol{\beta})]] \qquad (5)$$

Evaluating the rewritten Eq. 4 for a node pair sampled from $g$ gives a noisy but unbiased estimate of the ELBO. Following [15], the stochastic natural gradients computed from a sample pair $(a, b)$ are

$$\partial\gamma_{a,k}^t = \alpha_k + \frac{1}{g(a,b)}\phi_{a\to b,k}^t - \gamma_{a,k}^{t-1} \qquad (6)$$

$$\partial\lambda_{k,i}^t = \eta_{k,i} + \frac{1}{g(a,b)}\phi_{a\to b,k} \cdot \phi_{a\leftarrow b,k} \cdot y_{ab,i} - \lambda_{k,i}^{t-1}, \qquad (7)$$

where $y_{ab,0} = y_{ab}$, and $y_{ab,1} = 1 - y_{ab}$. In practice, we sample a "mini-batch" $S$ of pairs per update, to reduce noise.

The intuition behind the above update is analogous to Online LDA [15]. When a single pair $(a, b)$ is sampled, we are computing the setting of $\boldsymbol{\gamma}$ that would be optimal (given $\phi^t$) if our entire network were a multigraph consisting of the interaction between $a$ and $b$ repeated $1/g(a, b)$ times.

Our algorithm has assumed that the subset of node pairs $S$ are sampled independently. We can relax this assumption by defining a distribution over predefined sets of links. These sets can be defined using prior information about the pairs, such as network topology, which lets us take advantage of more sophisticated sampling strategies. For example, we can define a set for each node, with each set consisting of the node's adjacent links or non-links. Each iteration we set $S$ to one of these sets sampled at random from the $N$ sets.

In order to ensure that set-based sampling results in unbiased gradients, we specify two constraints on sets. First, we assume that the union of these sets $s$ is the total set of all node pairs, $U$: $U = \cup_i s_i$. Second, we assume that every pair $(a, b)$ occurs in some constant number of sets $c$ and that $c \geq 1$. With these conditions satisfied, we can again rewrite Eq. 4 as the sum over its global terms and an expectation over the local terms. Let $h(t)$ be a distribution over predefined sets of node pairs. For example, the fourth term in Eq. 4 can be rewritten using

$$\sum_{a,b} \mathbb{E}_q[\log p(y_{ab}|z_{a\to b}, z_{a\leftarrow b}, \boldsymbol{\beta})] = \mathbb{E}_h[\frac{1}{c}\frac{1}{h(t)}\sum_{(a,b)\in s_t} \mathbb{E}_q[\log p(y_{ab}|z_{a\to b}, z_{a\leftarrow b}, \boldsymbol{\beta})]] \qquad (8)$$

**Algorithm 1** Stochastic a-MMSB
---
1: Initialize $\boldsymbol{\gamma} = (\gamma_n)_{n=1}^N$, $\boldsymbol{\lambda} = (\lambda_k)_{k=1}^K$ randomly.
2: **while** convergence criteria is not met **do**
3:     Sample a subset $S$ of node pairs.
4:     **L-step**: Optimize $(\phi_{a \to b}, \phi_{a \leftarrow b})$ $\forall (a,b) \in S$
5:     Compute the natural gradients $\partial \gamma_n^t$ $\forall n$, $\partial \lambda_k^t$ $\forall k$
6:     **G-step**: Update $(\boldsymbol{\gamma}, \boldsymbol{\lambda})$ using Eq. 9.
7:     Set $\rho_t = (\tau_0 + t)^{-\kappa}$; $t \leftarrow t + 1$.
8: **end while**
---

The natural gradient of the random functions in Eq. 5 and Eq. 8 with respect to the global variational parameters $(\boldsymbol{\gamma}, \boldsymbol{\lambda})$ is a noisy but unbiased estimate of the natural gradient of the ELBO in Eq. 4.

However we subsample, the global step follows the noisy gradient with an appropriate step-size,

$$\boldsymbol{\gamma} \leftarrow \boldsymbol{\gamma} + \rho_t \partial \boldsymbol{\gamma}^t; \quad \boldsymbol{\lambda} \leftarrow \boldsymbol{\lambda} + \rho_t \partial \boldsymbol{\lambda}^t. \tag{9}$$

We require that $\sum_t \rho_t^2 < \infty$ and $\sum_t \rho_t = \infty$ for convergence to a local optimum [13]. We set $\rho_t \triangleq (\tau_0 + t)^{-\kappa}$, where $\kappa \in (0.5, 1]$ is the learning rate and $\tau_0 \geq 0$ downweights early iterations.

**The local step.** The local step optimizes the interaction parameters $\phi$ with respect to a subsample of the network. Recall that there is a per-interaction membership parameter for each node pair— $\phi_{a \to b}$ and $\phi_{a \leftarrow b}$—representing the posterior approximation of which communities are active in determining whether there is a link. We optimize these parameters in parallel. The update for $\phi_{a \to b}$ given $y_{a,b}$ is

$$\phi_{a \to b, k}^t | y = 0 \propto \exp\{\mathbb{E}_q[\log \pi_{a,k}] + \phi_{a \leftarrow b, k}^{t-1} \mathbb{E}_q[\log(1 - \beta_k)]$$

$$\phi_{a \to b, k}^t | y = 1 \propto \exp\{\mathbb{E}_q[\log \pi_{a,k}] + \phi_{a \leftarrow b, k}^{t-1} \mathbb{E}_q[\log \beta_k] + (1 - \phi_{a \leftarrow b, k}^{t-1}) \log \epsilon. \tag{10}$$

The updates for $\phi_{a \leftarrow b}$ are symmetric. This is natural gradient ascent with a step-size of one.

We present the full Stochastic a-MMSB algorithm in Algorithm 1. Each iteration subsamples the network and computes the local and global updates. We have derived this algorithm with node pairs sampled from arbitrary population distributions $g(a, b)$ or $h(t)$. One advantage of this approach is that we can explore various subsampling techniques without compromising the correctness of Algorithm 1. We will discuss and study sampling methods in §3.3 and §5. First, however, we discuss convergence and complexity.

**Held-out sets and convergence criteria.** We stop training on a network (the *training* set) when the average change in expected log likelihood on held-out data (the *validation* set) is less than 0.001%. The test and validation sets used in §5 have equal parts links and non-links, selected randomly from the network. A 50% links validation set poorly represents the severe class imbalance between links and non-links in real-world networks. However, a validation set matching the network sparsity would have too few links. Therefore, we compute the validation log likelihood at network sparsity by reweighting the average link and non-link log likelihood (estimated from the 50% links validation set) by their respective proportions in the network. We use a separate validation set to choose learning parameters and study sensitivity to $K$.

**Per-iteration complexity.** Our L-step can be computed in $O(nK)$, where $n$ is the number of node pairs sampled in each iteration. This is unlike MMSB, where the $\phi$ updates incur a cost quadratic in $K$. Step 6 requires that all nodes must be updated in each iteration. The time for a G-step in Algorithm 1 is $O(NK)$ and the total memory required is $O(NK)$.

### 3.3 Sampling strategies

Our algorithm allows us flexibility around how the subset of pairs is sampled, as long as the expectation of the stochastic gradient is equal to the true gradient. There are several ways we can take advantage of this. We can sample based on informative pairs of nodes, i.e., ones that help us better assess the community structure. We can also subsample to make data processing easier, for example, to accomodate a stream of links. Finally, large, real-world networks are often sparse, with links

accounting for less than 0.1% of all node pairs (see Figure 2). While we should not ignore non-links, it may help to give preferential attention to links. These intuitions are captured in the following four subsampling methods.

**Random pair sampling.** The simplest method is to sample node pairs uniformly at random. This method is an instance of independent pair sampling, with $g(a, b)$ (used in Eq. 5) equal to $\frac{1}{N(N-1)/2}$.

**Random node sampling.** This method focuses on local neighborhoods of the network. A set consists of all the pairs that involve one of the $N$ nodes. At each iteration, we sample a set uniformly at random from the $N$ sets, so $h(t)$ (used in Eq. 8) is $\frac{1}{N}$. Since each pair involves two nodes, each link appears in two sets, so $c$ (also used in Eq. 8) is 2. By reweighting the terms corresponding to pairs in the sampled set, we maintain a correct stochastic optimization.

**Stratified random pair sampling.** This method samples links independently, but focuses on observed links. We divide the $M$ node pairs into two *strata*: links and non-links. Each iteration either samples a mini-batch of links or samples a mini-batch of non-links. If the non-link stratum is sampled, and $N_0$ is the estimated total number of non-links, then

$$g(a, b) = \begin{cases} \frac{1}{N_0} & \text{if } y_{ab} = 0, \\ 0 & \text{if } y_{ab} = 1 \end{cases} \qquad (11)$$

The population distribution when the link strata is sampled is symmetric.

**Stratified random node sampling.** This method combines set-based sampling and stratified sampling to focus on observed links in local neighborhoods. For each node we define a "link set" consisting of all its links, and $m$ "non-link sets" that partition its non-links. Since the number of non-links associated with each node is usually large, dividing them into many sets allows the computation in each iteration to be fast. At each iteration, we first select a random node and either select its link set or sample one of its $m$ non-link sets, uniformly at random. To compute Eq. 8 we define the number of sets that contain each pair, $c = 2$, and the population distribution over sets

$$h(t) = \begin{cases} \frac{1}{2N} & \text{if } t \text{ is a link set,} \\ \frac{1}{2Nm} & \text{if } t \text{ is a non-link set.} \end{cases} \qquad (12)$$

Stratified random node sampling gives the best gains in convergence speed (see §5).

## 4 Related work

Newman et al. [3] described a model of overlapping communities in networks ("the Poisson model") where the number of links between two nodes is a Poisson random variable. Recently, other researchers have proposed latent feature network models [20, 21] that compute the probabilities of links based on the interactions between binary features associated with each node. Efficient inference algorithms for these models exploit model-specific approximations that allow scaling in the number of links. These ideas do not extend to the MMSB. Further, these algorithms do not explicitly leverage network sampling. In contrast, the ideas in Algorithm 1 apply to a number of models [14]. It subsamples both links and non-links in an inner loop for scalability.

Other scalable algorithms include Clique Percolation (CP) [7] and Link Clustering (LC) [8], which are based on heuristic clique-finding and hierarchical clustering, respectively. These methods are fast in practice, although the underlying problem is NP-complete. Further, because they are not statistical models, there is no clear mechanism for predicting new observations or model checking.

In the next section we will compare our method to these alternative scalable methods. Compared to the Poisson model, we will show that the MMSB gives better predictions. Compared to CP and LC, which do not provide predictions, we will show that the MMSB more reliably recovers the true community structure.

## 5 Empirical study

In this section, we evaluate the efficiency and accuracy of Stochastic a-MMSB (AM). First, we evaluate its efficiency on 10 real-world networks. Second, we demonstrate that stratified sampling

Figure 2: Network datasets. $N$ is the number of nodes, $K^{max}$ is the maximum number of communities analyzed and $d$ is the percent of node pairs that are links. The time until convergence for the different methods are $T_c^{stoch}$ and $T_c^{batch}$, while the time required for 90% of the perplexity at a-MMSB's convergence is $T_{90\%}^{stoch}$.

| DATA SET | $N$ | $K^{max}$ | $d(\%)$ | $T_{90\%}^{stoch}$ | $T_c^{stoch}$ | $T_c^{batch}$ | TYPE | SOURCE |
|---|---|---|---|---|---|---|---|---|
| US-AIR | 1.1K | 19 | 1.2 | $1.7m$ | $3.4m$ | $40.5m$ | TRANSP. | [22] |
| NETSCIENCE | 1.6K | 100 | 0.3 | $7.2m$ | $11.7m$ | $2.2h$ | COLLAB. | [16] |
| RELATIVITY | 5.2K | 300 | 0.1 | $2.3h$ | $4h$ | $>29h$ | COLLAB. | [23] |
| HEP-TH | 9.9K | 32 | 0.05 | $7.3h$ | $8.7h$ | $>67h$ | COLLAB. | [23] |
| HEP-PH | 12K | 32 | 0.16 | $36m$ | $2.8h$ | $>67h$ | COLLAB. | [23] |
| ASTRO-PH | 18.7K | 32 | 0.11 | $13.8h$ | $22.1h$ | $>67h$ | COLLAB. | [23] |
| HEP-TH2 | 27.8K | 512 | 0.09 | $8d$ | $10.3d$ | - | CITE | [23],[24] |
| ENRON | 37K | 158 | 0.03 | $1.5d$ | $2.5d$ | - | EMAIL | [25] |
| COND-MAT | 40.4K | 300 | 0.02 | $4.6d$ | $5.2d$ | - | COLLAB. | [26] |
| BRIGHTKITE | 58.2K | 64 | 0.01 | $8d$ | $9.5d$ | - | SOCIAL | [27] |

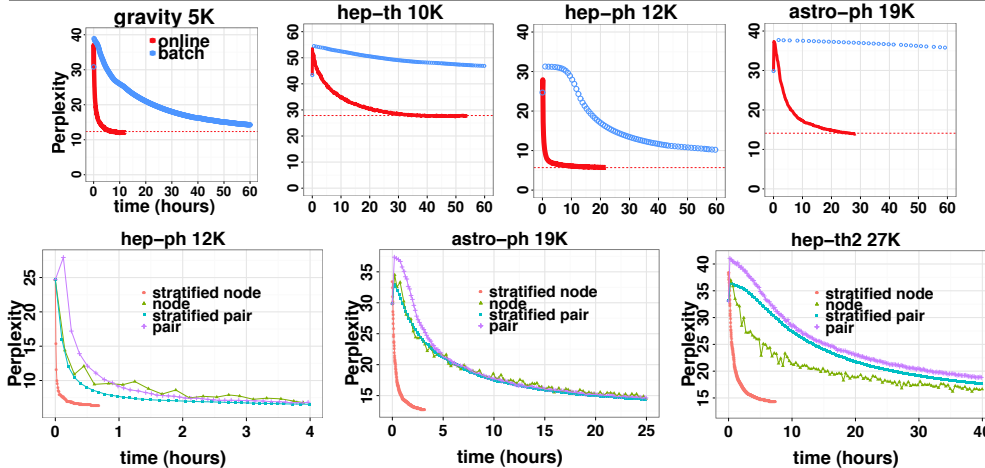

Figure 3: Stochastic a-MMSB (with random pair sampling) scales better and finds communities as good as batch a-MMSB on real networks (Top). Stratified random node sampling is an order of magnitude faster than other sampling methods on the hep-ph, astro-ph and hep-th2 networks (Bottom).

significantly improves convergence speed on real networks. Third, we compare our algorithm with leading algorithms in terms of accuracy on benchmark graphs and ability to predict links.

We measure convergence by computing the link prediction accuracy on a validation set. We set aside a validation and a test set, each having 10% of the network links and an equal number of non-links (see §3.2). We approximate the probability that a link exists between two nodes using posterior expectations of $\boldsymbol{\beta}$ and $\boldsymbol{\pi}$. We then calculate *perplexity*, which is the exponential of the average predictive log likelihood of held-out node pairs.

For random pair and stratified random pair sampling, we use a mini-batch size $S = N/2$ for graphs with $N$ nodes. For the stratified random node sampling, we set the number of non-link sets $m = 10$. Based on experiments, we set the parameters $\kappa = 0.5$ and $\tau_0 = 1024$. We set hyperparameters $\alpha = 1/K$ and $\{\eta_1, \eta_0\}$ proportional to the expected number of links and non-links in each community. We implemented all algorithms in C++.

**Comparing scalability to batch algorithms.** AM is an order of magnitude faster than standard batch inference for a-MMSB [2]. Figure 2 shows the time to convergence for four networks[3] of varying types, node sizes $N$ and sparsity $d$. Figure 3 shows test perplexity for batch vs. stochastic inference. For many networks, AM learns rapidly during the early iterations, achieving 90% of the converged perplexity in less than 70% of the full convergence time. For all but the two smallest networks, batch inference did not converge within the allotted time. AM lets us efficiently fit a mixed-membership model to large networks.

**Comparing sampling methods.** Figure 3 shows that stratified random node sampling converges an order of magnitude faster than random node sampling. It is statistically more efficient because the observations in each iteration include all the links of a node and a random sample of its non-links.

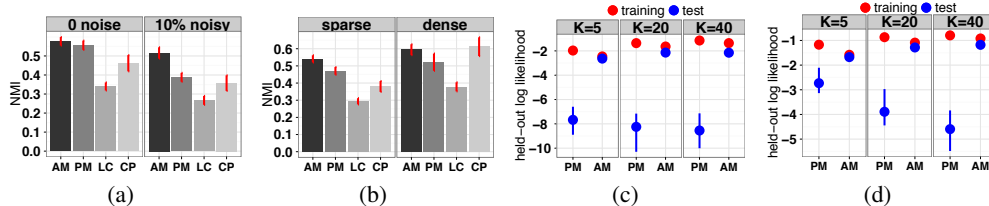

Figure 4: Figures (a) and (b) show that Stochastic a-MMSB (AM) outperforms the Poisson model (PM), Clique Percolation (CP), and Link Clustering (LC) in accurately recovering overlapping communities in 280 benchmark networks [28]. Each figure shows results on a binary partition of the 280 networks. Accuracy is measured using normalized mutual information (NMI) [28]; error bars denote the 95% confidence interval around the mean NMI. Figures (c) and (d) show that a-MMSB generalizes to new data better than PM on the netscience and us-air network, respectively. Each algorithm was run with 10 random initializations per $K$.

Figure 3 also shows that stratified random pair sampling converges ~1x–2x faster than random pair sampling.

**Comparing accuracy to scalable algorithms.** AM can recover communities with equal or better accuracy than the best scalable algorithms: the Poisson model (PM) [3], Clique percolation (CP) [7] and Link clustering (LC) [8]. We measure the ability of algorithms to recover overlapping communities in synthetic networks generated by the benchmark tool [28].[4] Our synthetic networks reflect real-world networks by modeling noisy links and by varying community densities from sparse to dense. We evaluate using normalized mutual information (NMI) between discovered communities and the ground truth communities [28]. We ran PM and a-MMSB until validation log likelihood changed by less than 0.001%. CP and LC are deterministic, but results vary between parameter settings. We report the best solution for each model.[5]

Figure 4 shows results for the 280 synthetic networks split in two ways. AM outperforms PM, LC, and CP on noisy networks and networks with sparse communities, and it matches the best performance in the noiseless case and the dense case. CP performs best on networks with dense communities—they tend to have more $k$-cliques—but with a larger margin of error than AM.

**Comparing predictive accuracy to PM.** Stochastic a-MMSB also beats PM [3], the best scalable probabilistic model of overlapping communities, in predictive accuracy. On two networks, we evaluated both algorithms' ability to predict held out links and non-links. We ran both PM and a-MMSB until their validation log likelihood changed less than 0.001%. Figures 4(c) and 4(d) show training and testing likelihood. PM overfits, while the a-MMSB generalizes well.

**Using the a-MMSB as an exploratory tool.** AM opens the door to large-scale exploratory analysis of real-world networks. In addition to the co-authorship network in Figure 1(a), we analyzed the "cond-mat" collaboration network [26] with the number of communities set to 300. This network contains 40,421 scientists and 175,693 links. In the supplement, we visualized the top authors in the network by a measure of their participation in different communities (*bridgeness* [17]). Finding such bridging nodes in a network is an important task in disease prevention and marketing.

### Acknowledgments

D.M. Blei is supported by ONR N00014-11-1-0651, NSF CAREER 0745520, AFOSR FA9550-09-1-0668, the Alfred P. Sloan foundation, and a grant from Google.

## Footnotes

[1]We use a subclass of the MMSB models that is appropriate for community detection in undirected networks. In particular, we assume *assortativity*, i.e., that links imply that nodes are similar. We call this special case the assortative MMSB or a-MMSB. In §2 we argue why the a-MMSB is more appropriate for community detection than the MMSB. We note that our algorithms are immediately applicable to the MMSB as well.

[2]Stochastic variational inference uses natural gradients [19] of the ELBO. Computing natural gradients (along with subsampling) leads to scalable variational inference algorithms [14].

[3]Following [1], we treat the directed citation network hep-th2 as an undirected network.

[4]We generated 280 networks for combinations of these parameters: #nodes$\in$ $\{400\}$; #communities$\in\{5, 10\}$; #nodes with at least 3 overlapping communities$\in\{100\}$; community sizes$\in$\{equal, unequal\}, when unequal, the community sizes are in the range $[\frac{N}{2K}, \frac{2N}{K}]$; average node degree$\in \{0.1\frac{N}{K}, 0.15\frac{N}{K}, .., 0.35\frac{N}{K}, 0.4\frac{N}{K}\}$, the maximum node degree=2×average node degree; % links of a node that are noisy$\in \{0, 0.1\}$; random runs$\in\{1,..,5\}$.

[5]CP finds a solution per clique size; LC finds a solution per threshold at which the dendrogram is cut [8] in steps of 0.1 from 0 to 1; PM and a-MMSB find a solution $\forall K \in \{k', k' + 10\}$ where $k'$ is the true number of communities—increasing by 10 allows for potentially a larger number of communities to be detected; a-MMSB also finds a solution for each of random pair or stratified random pair sampling methods with the hyperparameters $\eta$ set to the default or set to fit dense clusters.

# References

[1] Santo Fortunato. Community detection in graphs. *Physics Reports*, 486(35):75–174, 2010.

[2] E. Airoldi, D. Blei, S. Fienberg, and E. Xing. Mixed membership stochastic blockmodels. *Journal of Machine Learning Research*, 9:1981–2014, 2008.

[3] Brian Ball, Brian Karrer, and M. E. J. Newman. Efficient and principled method for detecting communities in networks. *Physical Review E*, 84(3):036103, 2011.

[4] M. E. J. Newman and M. Girvan. Finding and evaluating community structure in networks. *Physical Review E*, 69(2):026113, 2004.

[5] K. Nowicki and T. Snijders. Estimation and prediction for stochastic blockstructures. *Journal of the American Statistical Association*, 96(455):1077–1087, 2001.

[6] Peter J. Bickel and Aiyou Chen. A nonparametric view of network models and Newman-Girvan and other modularities. *Proceedings of the National Academy of Sciences*, 106(50):21068–21073, 2009.

[7] Imre Dernyi, Gergely Palla, and Tams Vicsek. Clique percolation in random networks. *Physical Review Letters*, 94(16):160202, 2005.

[8] Yong-Yeol Ahn, James P. Bagrow, and Sune Lehmann. Link communities reveal multiscale complexity in networks. *Nature*, 466(7307):761–764, 2010.

[9] M. E. J. Newman and E. A. Leicht. Mixture models and exploratory analysis in networks. *Proceedings of the National Academy of Sciences*, 104(23):9564–9569, 2007.

[10] A. Goldenberg, A. Zheng, S. Fienberg, and E. Airoldi. A survey of statistical network models. *Foundations and Trends in Machine Learning*, 2:129–233, 2010.

[11] W. Fu, L. Song, and E. Xing. Dynamic mixed membership blockmodel for evolving networks. In *ICML*, 2009.

[12] Qirong Ho, Ankur P. Parikh, and Eric P. Xing. A multiscale community blockmodel for network exploration. *Journal of the American Statistical Association*, 107(499):916–934, 2012.

[13] H. Robbins and S. Monro. A stochastic approximation method. *The Annals of Mathematical Statistics*, 22(3):400–407, 1951.

[14] M. Hoffman, D. Blei, C. Wang, and J. Paisley. Stochastic variational inference. *arXiv:1206.7051*, 2012.

[15] M. Hoffman, D. Blei, and F. Bach. Online learning for latent Dirichlet allocation. In *NIPS*, 2010.

[16] M. E. J. Newman. Finding community structure in networks using the eigenvectors of matrices. *Physical Review E*, 74(3):036104, 2006.

[17] Tams Nepusz, Andrea Petrczi, Lszl Ngyessy, and Flp Bazs. Fuzzy communities and the concept of bridgeness in complex networks. *Physical Review E*, 77(1):016107, 2008.

[18] M. Jordan, Z. Ghahramani, T. Jaakkola, and L. Saul. Introduction to variational methods for graphical models. *Machine Learning*, 37:183–233, 1999.

[19] S. Amari. Differential geometry of curved exponential families-curvatures and information loss. *The Annals of Statistics*, 1982.

[20] M. Morup, M.N. Schmidt, and L.K. Hansen. Infinite multiple membership relational modeling for complex networks. In *IEEE MLSP*, 2011.

[21] M. Kim and J. Leskovec. Modeling social networks with node attributes using the multiplicative attribute graph model. In *UAI*, 2011.

[22] RITA. U.S. Air Carrier Traffic Statistics, Bur. Trans. Stats, 2010.

[23] J. Leskovec, J. Kleinberg, and C. Faloutsos. Graph evolution: Densification and shrinking diameters. *ACM TKDD*, 2007.

[24] J. Gehrke, P. Ginsparg, and J. M. Kleinberg. Overview of the 2003 KDD cup. *SIGKDD Explorations*, 5:149–151, 2003.

[25] B. Klimmt and Y. Yang. Introducing the Enron corpus. In *CEAS*, 2004.

[26] M. E. J. Newman. The structure of scientific collaboration networks. *Proceedings of the National Academy of Sciences*, 98(2):404–409, 2001.

[27] J. Leskovec, K. J. Lang, A. Dasgupta, and M. W. Mahone. Community structure in large networks: Natural cluster sizes and the absence of large well-defined cluster. In *Internet Mathematics*, 2008.

[28] Andrea Lancichinetti and Santo Fortunato. Benchmarks for testing community detection algorithms on directed and weighted graphs with overlapping communities. *Physical Review E*, 80(1):016118, 2009.

